# Genetic Algorithms and Explicit Search Statistics

**Shumeet Baluja**
baluja@cs.cmu.edu
Justsystem Pittsburgh Research Center &
School of Computer Science, Carnegie Mellon University

## Abstract

The genetic algorithm (GA) is a heuristic search procedure based on mechanisms abstracted from population genetics. In a previous paper [Baluja & Caruana, 1995], we showed that much simpler algorithms, such as hillclimbing and Population-Based Incremental Learning (PBIL), perform comparably to GAs on an optimization problem custom designed to benefit from the GA's operators. This paper extends these results in two directions. First, in a large-scale empirical comparison of problems that have been reported in GA literature, we show that on many problems, simpler algorithms can perform significantly better than GAs. Second, we describe when crossover is useful, and show how it can be incorporated into PBIL.

## 1 IMPLICIT VS. EXPLICIT SEARCH STATISTICS

Although there has recently been controversy in the genetic algorithm (GA) community as to whether GAs should be used for static function optimization, a large amount of research has been, and continues to be, conducted in this direction [De Jong, 1992]. Since much of GA research focuses on optimization (most often in static environments), this study examines the performance of GAs in these domains.

In the standard GA, candidate solutions are encoded as fixed length binary vectors. The initial group of potential solutions is chosen randomly. At each generation, the fitness of each solution is calculated; this is a measure of how well the solution optimizes the objective function. The subsequent generation is created through a process of selection, recombination, and mutation. Recombination operators merge the information contained within pairs of selected "parents" by placing random subsets of the information from both parents into their respective positions in a member of the subsequent generation. The fitness proportional selection works as selective pressure; higher fitness solution strings have a higher probability of being selected for recombination. Mutations are used to help preserve diversity in the population by introducing random changes into the solution strings. The GA uses the population to *implicitly* maintain statistics about the search space. The selection, crossover, and mutation operators can be viewed as mechanisms of extracting the implicit statistics from the population to choose the next set of points to sample. Details of GAs can be found in [Goldberg, 1989] [Holland, 1975].

Population-based incremental learning (PBIL) is a combination of genetic algorithms and competitive learning [Baluja, 1994]. The PBIL algorithm attempts to *explicitly* maintain statistics about the search space to decide where to sample next. The object of the algorithm is to create a real valued probability vector which, when sampled, reveals high quality solution vectors with high probability. For example, if a good solution can be encoded as a string of alternating 0's and 1's, a suitable final probability vector would be 0.01, 0.99, 0.01, 0.99, etc. The PBIL algorithm and parameters are shown in Figure 1.

Initially, the values of the probability vector are initialized to 0.5. Sampling from this vector yields random solution vectors because the probability of generating a 1 or 0 is equal. As search progresses, the values in the probability vector gradually shift to represent high

```
****** Initialize Probability Vector *****
for i :=1 to LENGTH do P[i] = 0.5;

while (NOT termination condition)
          ***** Generate Samples *****
          for i :=1 to SAMPLES do
                    sample_vectors[i] := generate_sample_vector_according_to_probabilities (P);
                    evaluations[i] := evaluate(sample_vectors[i]);

          best_vector := find_vector_with_best_evaluation (sample_vectors, evaluations);
          worst_vector := find_vector_with_worst_evaluation (sample_vectors, evaluations);

          ***** Update Probability Vector Towards Best Solution *****
          for i :=1 to LENGTH do
                    P[i] := P[i] * (1.0 - LR) + best_vector[i] * (LR);

          ***** Update Probability Vector Away from Worst Solution *****
          for i :=1 to LENGTH do
                    if (best_vector[i] ≠ worst_vector[i]) then
                              P[i] := P[i] * (1.0 - NEGATIVE_LR) + best_vector[i] * (NEGATIVE_LR);
```

**PBIL: USER DEFINED CONSTANTS (Values Used in this Study):**
SAMPLES: the number of vectors generated before update of the probability vector (100).
LR: the learning rate, how fast to exploit the search performed (0.1).
NEGATIVE_LR: negative learning rate, how much to learn from negative examples (PBIL1=0.0, PBIL2= 0.075).
LENGTH: the number of bits in a generated vector (problem specific).

**Figure 1:** PBIL1/PBIL2 algorithm for a binary alphabet. PBIL2 includes shaded region. Mutations not shown.

evaluation solution vectors through the following process. A number of solution vectors are generated based upon the probabilities specified in the probability vector. The probability vector is pushed towards the generated solution vector with the highest evaluation. After the probability vector is updated, a new set of solution vectors is produced by sampling from the updated probability vector, and the cycle is continued. As the search progresses, entries in the probability vector move away from their initial settings of 0.5 towards either 0.0 or 1.0.

One key feature of the *early* generations of genetic optimization is the parallelism in the search; many diverse points are represented in the population of points during the early generations. When the population is diverse, crossover can be an effective means of search, since it provides a method to explore novel solutions by combining different members of the population. Because PBIL uses a single probability vector, it may seem to have less expressive power than a GA using a full population, since a GA can *represent* a large number of points simultaneously. A traditional single population GA, however, would not be able to *maintain* a large number of points. Because of sampling errors, the population will converge around a single point. This phenomenon is summarized below:

> "... the theorem [Fundamental Theorem of Genetic Algorithms [Goldberg, 1989]], assumes an infinitely large population size. In a finite size population, even when there is no selective advantage for either of two competing alternatives... the population will converge to one alternative or the other in finite time (De Jong, 1975; [Goldberg & Segrest, 1987]). This problem of finite populations is so important that geneticists have given it a special name, genetic drift. Stochastic errors tend to accumulate, ultimately causing the population to converge to one alternative or another" [Goldberg & Richardson, 1987].

Diversity in the population is crucial for GAs. By maintaining a population of solutions, the GA is able—in theory at least—to maintain samples in many different regions. Crossover is used to merge these different solutions. A necessary (although not sufficient) condition for crossover to work well is diversity in the population. When diversity is lost, crossover begins to behave like a mutation operator that is sensitive to the convergence of the value of each bit [Eshelman, 1991]. If all individuals in the population converge at

some bit position, crossover leaves those bits unaltered. At bit positions where individuals have not converged, crossover will effectively mutate values in those positions. Therefore, crossover creates new individuals that differ from the individuals it combines only at the bit positions where the mated individuals disagree. This is analogous to PBIL which creates new trials that differ mainly in positions where prior good performers have disagreed.

As an example of how the PBIL algorithm works, we can examine the values in the probability vector through multiple generations. Consider the following maximization problem: $1.0/|(366503875925.0 - X)|$, $0 \leq X < 2^{40}$. Note that 366503875925 is represented in binary as a string of 20 pairs of alternating '01'. The evolution of the probability vector is shown in Figure 2. Note that the most significant bits are pinned to either 0 or 1 very quickly, while the least significant bits are pinned last. This is because during the early portions of the search, the most significant bits yield more information about high evaluation regions of the search space than the least significant bits.

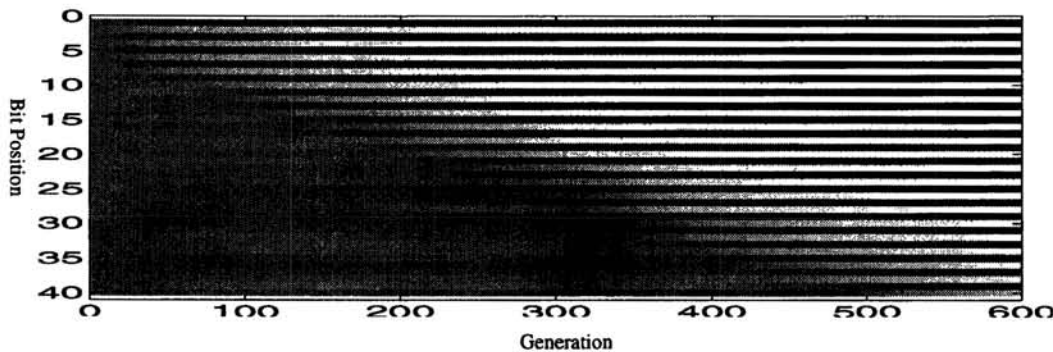

**Figure 2:** Evolution of the probability vector over successive generations. White represents a high probability of generating a 1, black represents a high probability of generating a 0. Intermediate grey represent probabilities close to 0.5 - equal chances of generating a 0 or 1. Bit 0 is the most significant, bit 40 the least.

## 2  AN EMPIRICAL COMPARISON

This section provides a summary of the results obtained from a large scale empirical comparison of seven iterative and evolution-based optimization heuristics. Thirty-four static optimization problems, spanning six sets of problem classes which are commonly explored in the genetic algorithm literature, are examined. The search spaces in these problems range from $2^{128}$ to $2^{2040}$. The results indicate that, on many problems, using standard GAs for optimizing static functions does not yield a benefit, in terms of the final answer obtained, over simple hillclimbing or PBIL. Recently, there have been other studies which have examined the performance of GAs in comparison to hillclimbing on a few problems; they have shown similar results [Davis, 1991][Juels & Wattenberg, 1996].

Three variants of Multiple-Restart Stochastic Hillclimbing (MRSH) are explored in this paper. The first version, MRSH-1, maintains a list of the position of the bit flips which were attempted without improvement. These bit flips are not attempted again until a better solution is found. When a better solution is found, the list is emptied. If the list becomes as large as the solution encoding, MRSH-1 is restarted at a random solution with an empty list. MRSH-2 and MRSH-3 allow moves to regions of higher and equal evaluation. In MRSH-2, the number of evaluations before restart depends upon the length of the encoded solution. MRSH-2 allows 10*(length of solution) evaluations without improvement before search is restarted. When a solution with a higher evaluation is found, the count is reset. In MRSH-3, after the total number of iterations is specified, restart is forced 5 times during search, at equally spaced intervals.

Two variants of the standard GA are tested in this study. The first, termed SGA, has the following parameters: Two-Point crossover, with a crossover rate of 100% (% of times crossover occurs, otherwise the individuals are copied without crossover), mutation probability of 0.001 per bit, population size of 100, and elitist selection (the best solution in

generation N replaces the worst solution in generation N+1). The second GA used, termed GA-Scale, uses the same parameters except: uniform crossover with a crossover rate of 80% and the fitness of the worst member in a generation is subtracted from the fitnesses of each member of the generation before the probabilities of selection are determined.

Two variants of PBIL are tested. Both move the probability vector towards the best example in each generated population. PBIL2 also moves the probability vector away from the worst example in each generation. Both variants are shown in Figure 1. A small mutation, analogous to the mutation used in genetic algorithms, is also used in both PBILs. The mutation is directly applied to the probability vector.

The results obtained in this study should *not* be considered to be state-of-the-art. The problem encodings were chosen to be easily reproducible and to allow easy comparison with other studies. Alternate encodings may yield superior results. In addition, no problem-specific information was used for any of the algorithms. Problem-specific information, when available, could help all of the algorithms examined.

All of the variables in the problems were encoded in binary, either with standard Gray-code or base-2 representation. The variables were represented in non-overlapping, contiguous regions within the solution encoding. The results reported are the best evaluations found through the search of each algorithm, averaged over at least 20 independent runs per algorithm per problem; the results for GA-SCALE and PBIL2 algorithms are the average of at least 50 runs. All algorithms were given 200,000 evaluations per run. In each run, the GA and PBIL algorithms were given 2000 generations, with 100 function evaluations per generation. In each run, the MRSH algorithms were restarted in random locations as many times as needed until 200,000 evaluations were performed. The best answer found in the 200,000 evaluations was returned as the answer found in the run.

Brief notes about the encodings are given below. Since the numerical results are not useful without the exact problems, *relative* results are provided in Table I. For most of the problems, exact results and encodings are in [Baluja, 1995]. To measure the significance of the difference between the results obtained by PBIL2 and GA-SCALE, the Mann-Whitney test is used. This is a non-parametric equivalent to the standard two-sample pooled *t*-tests.

- **TSP:**       128, 200 & 255 city problems were tried. The "sort" encoding [Syswerda, 1989] was used. The last problem was tried with the encoding in binary and Gray-Code.

- **Jobshop:**   Two standard JS problems were tried with two encodings. The first encoding is described in [Fang *et. al*, 1993]. The second encoding is described in [Baluja, 1995]. An additional, randomly generated, problem was also tried with the second encoding.

- **Knapsack:** Problem 1&2: a unique element is represented by each bit. Problem 3&4: there are 8 and 32 copies of each element respectively. The encoding specified the number of copies of each element to include. Each element is assigned a "value" and "weight". Object: maximize value while staying under pre-specified weight.

- **Bin-Packing/Equal Piles:** The solution is encoded in a bit vector of length $M * \log_2 N$ (N bins, M elem.). Each element is assigned a substring of length $\log_2 N$, which specifies a bin. Object: pack the given bins as tightly as possible. Because of the large variation in results which is found by varying the number of bins and elements, the results from 8 problems are reported.

- **Neural-Network Weight Optimization:** Problem 1&2: identify the parity of 7 inputs. Problem 3&4: determine whether a point falls within the middle of 3 concentric squares. For problems 3&4, 5 extra inputs, which contained noise, were used. The networks had 8 inputs (including bias), 5 hidden units, and 1 output. The network was fully connected between sequential layers.

- **Numerical Function Optimization (F1-F3):** Problems 1&2: the variables in the first portions of the solution string have a large influence on the quality of the rest of the solution. In the third problem, each variable can be set independently. See [Baluja, 1995] for details.

- **Graph Coloring:** Select 1 of 4 colors for nodes of a partially connected graph such that connected nodes are not the same color. The graphs used were not necessarily planar.

**Table I: Summary of Empirical Results - Relative Ranks (1=best, 7=worst).**

| | Encoding Length (bits) | MRSH 1 | MRSH 2 | MRSH 3 | PBIL 1 | PBIL 2 | SGA | GA Scale | MRSH BEST | PBIL BEST | GA BEST | Confidence (GA-Scale ≠ PBIL2) |
|---|---|---|---|---|---|---|---|---|---|---|---|---|
| TSP 128 city (binary) | 896 | 6 | 3 | 4 | 2 | 1 | 7 | 5 | | • | | > 99% |
| TSP 200 city (binary) | 1600 | 5 | 4 | 3 | 2 | 1 | 7 | 6 | | • | | > 99% |
| TSP 255 city (binary) | 2040 | 5 | 1 | 2 | 4 | 3 | 7 | 6 | • | | | > 99% |
| TSP 255 city (Gray-Code) | 2040 | 5 | 1 | 2 | 4 | 3 | 7 | 6 | • | | | > 99% |
| Jobshop 10x10 (Fang) | 500 | 7 | 5 | 6 | 2 | 1 | 4 | 3 | | • | | > 99% |
| Jobshop 20x5 (Fang) | 500 | 7 | 6 | 5 | 2 | 1 | 4 | 3 | | • | | > 99% |
| Jobshop 10x10 (Baluja) | 700 | 7 | 5 | 6 | 3 | 1 | 4 | 2 | | • | | 93% |
| Jobshop 20x5 (Baluja) | 700 | 7 | 5 | 4 | 2 | 1 | 6 | 3 | | • | | > 99% |
| Jobshop 20x5 - random. (Baluja) | 700 | 7 | 5 | 4 | 2 | 1 | 6 | 3 | | • | | > 99% |
| Knapsack (512 elem., 1 copy) | 512 | 5 | 7 | 6 | 2 | 1 | 4 | 3 | | • | | > 99% |
| Knapsack (2000 elem., 1 copy) | 2000 | 4 | 5 | 6 | 1 | 3 | 7 | 2 | | • | | > 99% |
| Knapsack (100 elem., 8 copies) | 300 | 4 | 5 | 6 | 3 | 2 | 7 | 1 | | | • | Not Avail. |
| Knapsack (120 elem., 32 copies) | 600 | 4 | 5 | 6 | 2 | 1 | 7 | 3 | | • | | > 99% |
| Bin (2 bins, 1600 elements) | 1600 | 1 | 5 | 6 | 3 | 4 | 7 | 2 | • | | | 97% |
| Bin (2 bins, 128 elements) | 128 | 1 | 3 | 7 | 4 | 2 | 5 | 6 | • | | | > 99% |
| Bin (4 bins, 512 elements) | 1024 | 4 | 3 | 5 | 6 | 7 | 2 | 1 | | | • | > 99% |
| Bin (8 bins, 128 elements) | 384 | 6 | 5 | 7 | 2 | 1 | 4 | 3 | | • | | > 99% |
| Bin (16 bins,128 elements) | 512 | 7 | 5 | 6 | 3 | 1 | 2 | 4 | | • | | > 99% |
| Bin (32 bins, 128 elements) | 640 | 5 | 6 | 7 | 3 | 1 | 4 | 2 | | • | | 96% |
| Bin (32 bins, 256 elements) | 1280 | 2 | 4 | 5 | 6 | 7 | 3 | 1 | | | • | > 99% |
| Bin (64 bins, 128 elements) | 768 | 4 | 6 | 7 | 3 | 1 | 5 | 2 | | • | | > 99% |
| Neural Net PARITY 7 (binary) | 368 | 5 | 5 | 7 | 2 | 1 | 4 | 3 | | • | | > 99% |
| Neural Net PARITY 7 (gray) | 368 | 5 | 6 | 7 | 2 | 1 | 3 | 4 | | • | | > 99% |
| Neural Net SQUARE (binary) | 368 | 5 | 6 | 7 | 2 | 1 | 3 | 4 | | • | | > 99% |
| Neural Net SQUARE (gray) | 368 | 4 | 2 | 7 | 1 | 3 | 5 | 6 | | • | | > 99% |
| F1 (Encoded in Binary) | 900 | 5 | 6 | 7 | 3 | 1 | 2 | 4 | | • | | > 99% |
| F1 (Encoded in Gray Code) | 900 | 5 | 6 | 7 | 2 | 1 | 3 | 4 | | • | | > 99% |
| F2 (Encoded in Binary) | 900 | 5 | 6 | 7 | 2 | 1 | 4 | 3 | | • | | > 99% |
| F2 (Encoded in Gray Code) | 900 | 5 | 4 | 6 | 2 | 1 | 7 | 3 | | • | | > 99% |
| F3 (Encoded in Binary) | 900 | 6 | 5 | 7 | 2 | 1 | 4 | 3 | | • | | > 99% |
| F3 (Encoded in Gray Code) | 900 | 1 | 1 | 1 | 5 | 4 | 7 | 6 | • | | | > 99% |
| G.Color - 200 node, 1000 connx. | 400 | 6 | 2 | 1 | 4 | 3 | 7 | 5 | • | | | > 99% |
| G.Color - 200 node, 2000 connx. | 400 | 6 | 1 | 2 | 5 | 3 | 7 | 4 | • | | | > 99% |
| G.Color - 400 node, 8000 connx. | 800 | 5 | 1 | 2 | 4 | 3 | 7 | 6 | • | | | > 99% |
| TOTAL (34 Problems) | | | | | | | | | 8 | 23 | 3 | |

# 3  EXPLICITLY PRESERVING DIVERSITY

Although the results in the previous section showed that PBIL often outperformed GAs and hillclimbing, PBIL may not surpass GAs at all population sizes. As the population size increases, the observed behavior of a GA more closely approximates the ideal behavior predicted by theory [Holland, 1975]. The population may contain sufficient samples from distinct regions for crossover to effectively combine "building blocks" from multiple solutions. However, the desire to minimize the total number of function evaluations often prohibits the use of large enough populations to make crossover behave ideally.

One method of avoiding the cost of using a very large population is to use a parallel GA (pGA). Many studies have found pGAs to be very effective for preserving diversity for function optimization [Cohoon et al., 1988][Whitley et al., 1990]. In the pGA, a collection of independent GAs, each maintaining separate populations, communicate with each other

via infrequent inter-population (as opposed to intra-population) matings. pGAs suffer less from premature convergence than single population GAs. Although the individual populations typically converge, different populations converge to different solutions, thus preserving diversity across the populations. Inter-population mating permits crossover to combine solutions found in different regions of the search space.

We would expect that employing multiple PBIL evolutions, parallel PBIL (pPBIL), has the potential to yield performance improvements similar to those achieved in pGAs. Multiple PBIL evolutions are simulated by using multiple probability vectors to generate solutions. To keep the evolutions independent, each probability vector is only updated with solutions which are generated by sampling it.

The benefit of parallel populations (beyond just multiple runs) is in using crossover to combine dissimilar solutions. There are many ways of introducing crossover into PBIL. The method which is used here is to sample two probability vectors for the creation of each solution vector, see Figure 3. The figure shows the algorithm with uniform crossover; nonetheless, many other crossover operators can be used.

The randomized nature of crossover often yields unproductive results. If crossover is to be used, it is important to simulate the crossover operation many times. Therefore, crossover is used to create each member of the population (this is in contrast to crossing over the probability vectors once, and generating the entire population from the newly created probability vector). More details on integrating crossover and PBIL, and its use in combinatorial problems in robotic surgery can be found in [Baluja & Simon, 1996].

Results with using pPBIL in comparison to PBIL, GA, and pGA are shown in Table II. For many of the problems explored here, parallel versions of GAs and PBIL work better than the sequential versions, and the parallel PBIL models work better than the parallel GA models. In each of these experiments, the parameters were hand-tuned for each algorithms. *In every case, the GA was given at least twice as many function evaluations as PBIL.* The crossover operator was chosen by trying several operators on the GA, and selecting the best one. The same crossover operator was then used for PBIL. For the pGA and pPBIL experiments, 10 subpopulations were always used.

```
***** Generate Samples With Two Probability Vectors *****
for i :=1 to SAMPLES do
    vector1 := generate_sample_vector_with_probabilities (P1);
    vector2 := generate_sample_vector_with_probabilities (P2);
    for j := 1 to LENGTH_do
        if (random (2) = 0) sample_vector[i][j] := vector1[j]
        else   sample_vector[i][j] := vector2[j]
    evaluations[i] := Evaluate_Solution (sample[i]);
    best_vector := best_evaluation (sample_vectors, evaluations);

***** Update Both Probability Vectors Towards Best Solution *****
for i :=1 to LENGTH do
    P1[i] := P1[i] * (1.0 - LR) + best_vector[i] * (LR);
    P2[i] := P2[i] * (1.0 - LR) + best_vector[i] * (LR);
```

**Figure 3:** Generating samples based on two probability vectors. Shown with uniform crossover [Syswerda, 1989] (50% chance of using probability vector 1 or vector 2 for each bit position). Every 100 generations, each population makes a local copy of another population's probability vector (to replace vector2). In these experiments, there are a total of 10 subpopulations.

### Table II: Sequential & Parallel, GA & PBIL, Avg. 25 runs

| Problem (Minimize or Maximize Solution) | GA | pGA | PBIL | pPBIL |
|---|---|---|---|---|
| TSP - 128 city (minimize tour length) | 3256 | 2832 | 1718 | 1344 |
| TSP - 200 city (minimize tour length) | 14501 | 11633 | 6993 | 5012 |
| Numerical Optim. Highly Correlated Parameters - Base-2 Code (max) | 0.15 | 0.30 | 0.19 | 0.30 |
| Numerical Optim. Highly Correlated Parameters - Gray Code (max) | 0.18 | 0.31 | 0.18 | 1.6 |
| Numerical Optim. Independent Parameters - Base-2 Code (max) | 0.68 | 2.91 | 0.71 | 4.45 |
| Numerical Optim. Independent Parameters - Gray Code (max) | 8.33 | 8.33 | 8.33 | 8.33 |
| Checkerboard (Problem with many maxima, see [Baluja, 1994]) (max) | 1119 | 1150 | 1206 | 1256 |

# 4 SUMMARY & CONCLUSIONS

PBIL was examined on a very large set of problems drawn from the GA literature. The effectiveness of PBIL for finding good solutions for static optimization functions was compared with a variety of GA and hillclimbing techniques. Second, Parallel-PBIL was introduced. pPBIL is designed to explicitly preserve diversity by using multiple parallel evolutions. Methods for reintroducing crossover into pPBIL were given.

With regard to the empirical results, it should be noted that it is incorrect to say that one procedure will always perform better than another. The results *do not* indicate that PBIL will always outperform a GA. For example, we have presented problems on which GAs work better. Further, on problems such as binpacking, the relative results can change drastically depending upon the number of bins and elements. The conclusion which should be reached from these results is that algorithms, like PBIL and MRSH, which are much simpler than GAs, can outperform standard GAs on many problems of interest.

The PBIL algorithm presented here is very simple and should serve as a prototype for future study. Three directions for future study are presented here. First, the most obvious extension to PBIL is to track more detailed statistics, such as pair-wise covariances of bit positions in high-evaluation vectors. Preliminary work in this area has been conducted, and the results are very promising. Second, another extension is to quickly determine which probability vectors, in the pPBIL model, are unlikely to yield promising answers; methods such as Hoeffding Races may be adapted here [Maron & Moore, 1994]. Third, the manner in which the updates to the probability vector occur is similar to the weight update rules used in Learning Vector Quantization (LVQ). Many of the heuristics used in LVQ can be incorporated into the PBIL algorithm.

Perhaps the most important contribution of the PBIL algorithm is a novel way of examining GAs. In many previous studies of the GA, the GA was examined at a micro-level, analyzing the preservation of building blocks and frequency of sampling hyperplanes. In this study, the statistics at the population level were examined. In the standard GA, the population serves to *implicitly* maintain statistics about the search space. The selection and crossover mechanisms are ways of extracting these statistics from the population. PBIL's population does not maintain the information that is carried from one generation to the next. The statistics of the search are *explicitly* kept in the probability vector.

## References

Baluja, S. (1995) "An Empirical Comparison of Seven Iterative and Evolutionary Function Optimization Heuristics," CMU-CS-95-193. Available via. http://www.cs.cmu.edu/~baluja.

Baluja, S. (1994) "Population-Based Incremental Learning". Carnegie Mellon University. Technical Report. CMU-CS-94-163.

Baluja, S. & Caruana, R. (1995) "Removing the Genetics from the Standard Genetic Algorithm", *Inter.Conf. Mach. Learning-12*.

Baluja, S. & Simon, D. (1996) "Evolution-Based Methods for Selecting Point Data for Object Localization: Applications to Computer Assisted Surgery". CMU-CS-96 -183.

Cohoon, J., Hedge, S., Martin, W., Richards, D., (1988) "Distributed Genetic Algorithms for the Floor Plan Design Problem," School of Engineering and Applied Science, Computer Science Dept., University of Virginia, TR-88-12.

Davis, L.J. (1991) "Bit-Climbing, Representational Bias and Test Suite Design". *International Conf. on Genetic Algorithms 4*.

De Jong, K. (1975) *An Analysis of the Behavior of a Class of Genetic Adaptive Systems*. Ph.D. Dissertation.

De Jong, K. (1993) "Genetic Algorithms are NOT Function Optimizers". In Whitley (ed.) *Foundations of GAs-2*. 5-17.

Eshelman, L.J. (1991) "The CHC Adaptive Search Algorithm," in Rawlings (ed.) *Foundations of GAs-1*. 265-283.

Fang, H.L, Ross, P., Corne, D. (1993) "A Promising Genetic Algorithm Approach to Job-Shop Scheduling, Rescheduling, and Open- Shop Scheduling Problems". In Forrest, S. *International Conference on Genetic Algorithms 5*.

Goldberg, D.E. (1989) *Genetic Algorithms in Search, Optimization, and Machine Learning*. Addison-Wesley.

Goldberg & Richardson (1987) "Genetic Algorithms with Sharing for Multimodal Function Optimization" - *Proceedings of the Second International Conference on Genetic Algorithms*.

Holland, J. H. (1975) *Adaptation in Natural and Artificial Systems*. Ann Arbor: The University of Michigan Press.

Juels, A. & Wattenberg, M. (1994) "Stochastic Hillclimbing as a Baseline Method for Evaluating Genetic Algorithms" *NIPS 8*.

Maron, O. & Moore, A.(1994) "Hoeffding Races:Accelerating Model Selection for Classification and Function Approx." *NIPS 6*

Mitchell, M., Holland, J. & Forrest, S. (1994) "When will a Genetic Algorithm Outperform Hill Climbing" *NIPS 6*.

Syswerda, G. (1989) "Uniform Crossover in Genetic Algorithms," *International Conference on Genetic Algorithms 3*. 2-9.

Whitley, D., & Starkweather, T. "Genitor II: A Distributed Genetic Algorithm". *JETAI* 2: 189-214.
